# A Switched Gaussian Process for Estimating Disparity and Segmentation in Binocular Stereo

**Oliver Williams**
Microsoft Research Ltd.
Cambridge, UK
`omcw2@cam.ac.uk`

## Abstract

This paper describes a Gaussian process framework for inferring pixel-wise disparity and bi-layer segmentation of a scene given a stereo pair of images. The Gaussian process covariance is parameterized by a foreground-background-occlusion segmentation label to model both smooth regions and discontinuities. As such, we call our model a *switched Gaussian process*. We propose a greedy incremental algorithm for adding observations from the data and assigning segmentation labels. Two observation schedules are proposed: the first treats scanlines as independent, the second uses an active learning criterion to select a sparse subset of points to measure. We show that this probabilistic framework has comparable performance to the state-of-the-art.

## 1 Introduction

Given two views of the same scene, this paper addresses the dual objectives of inferring depth and segmentation in scenes with perceptually distinct foreground and background layers. We do this in a probabilistic framework using a Gaussian process prior to model the geometry of typical scenes of this type. Our approach has two properties of interest to practitioners: firstly, it can be employed incrementally which is useful for circumstances in which the time allowed for processing is constrained or variable; secondly it is probabilistic enabling fusion with other sources of scene information.

Segmentation and depth estimation are well-studied areas (e.g., [1] and [2, 3, 4]). However the inspiration for the work in this paper is [5] in which both segmentation and depth are estimated in a unified framework based around graph cuts. In [5] the target application was video conferencing, however such an algorithm is also applicable to areas such as robotics and augmented reality. Gaussian process regression has previously been used in connection with stereo images in [6] to learn the non-linear mapping between matched left-right image points and scene points as an alternative to photogrammetric camera calibration [7]. In this paper we use a Gaussian process to help discover the initially unknown left-right matches in a complex scene: a camera calibration procedure might then be used to determine actual 3D scene geometry.

The paper is organized as follows: Sec. 2 describes our Gaussian process framework for inferring depth (disparity) and segmentation from stereo measurements. Sec. 3 proposes and demonstrates two observation schedules: the first operates along image scanlines independently, the second treats the whole image jointly, and makes a sparse set of stereo observations at locations selected by an active learning criterion [8]; we also show how colour information may be fused with predictions by the switched GP, the results of which are comparable to those of [5]. Sec. 4 concludes the paper.

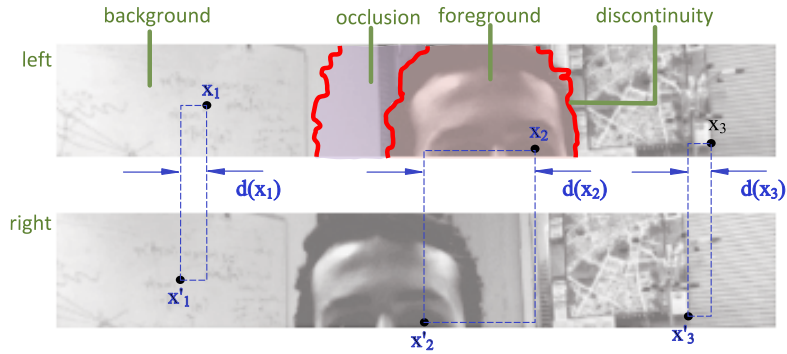

Figure 1: **Anatomy of a disparity map.** *This schematic shows some of the important features in short baseline binocular stereo for an horizontal strip of pixels. Transitions between foreground and background at the right edge of a foreground object will induce a* discontinuity *from high to low disparity. Background–foreground transitions at the left edge of the foreground induce an* occlusion region *in which scene points visible in the left image are not visible in the right. We use the data from [5] which are available on their web site: http://research.microsoft.com/vision/cambridge/i2i*

## 2   Single frame disparity estimation

This framework is intended for use with *short baseline* stereo, in which the two images are taken slightly to the left and the right of a midpoint (see Fig. 1). This means that most features visible in one image are visible in the other, albeit at a different location: for a given point $\mathbf{x}$ in the left image $L(\mathbf{x})$, our aim is therefore to infer the location of the same *scene* point in the right image $R(\mathbf{x}')$. We assume that both $L$ and $R$ have been *rectified* [7] such that all corresponding points have the same vertical coordinate; hence if $\mathbf{x} = [x\ y]^\mathrm{T}$ then $\mathbf{x}' = [x - d(\mathbf{x})\ y]^\mathrm{T}$ where $d(\mathbf{x})$ is called the *disparity map* for points $\mathbf{x}$ in the left image.

Because objects typically have smooth variations in depth, $d(\mathbf{x})$ is generally smooth. However, there are two important exceptions to this and, because they occur at the boundaries between an object and the background, it is essential that they be modelled correctly (see also Fig. 1):

**Discontinuity** Discontinuities occur where one pixel belongs to the foreground and its neighbour belongs to the background.

**Occlusion** At background–foreground transitions (travelling horizontally from left to right), there will be a region of pixels in the left image that are not visible in the right since they are *occluded* by the foreground [3]. Such locations correspond to scene points in the background layer, however their disparity is undefined.

The next subsection describes a prior for disparity that attempts to capture these characteristics by modelling the bi-layer segmentation.

### 2.1   A Gaussian process prior for disparity

We model the prior distribution of a disparity map to be a Gaussian process (GP) [9]. GPs are defined by a mean function $f(\cdot)$ and a covariance function $c(\cdot, \cdot)$ which in turn define the joint distribution of disparities at a set of points $\{\mathbf{x}_i, \ldots, \mathbf{x}_n\}$ as a multivariate Gaussian

$$P\big(d(\mathbf{x}_i), \ldots, d(\mathbf{x}_n)|f, c\big) = \text{Normal}\,(\mathbf{f},\ C) \tag{1}$$

where $\mathbf{f}_i = f(\mathbf{x}_i)$ and $C_{ij} = c(\mathbf{x}_i, \mathbf{x}_j)$.

In order to specify a mean and covariance function that give typical disparity maps an high probability, we introduce a latent segmentation variable $s(\mathbf{x}) \in \{\mathsf{F}, \mathsf{B}, \mathsf{O}\}$ for each point in the left image. This encodes whether a point belongs to the foreground (F), background (B) or is occluded (O) and makes it possible to model the fact that disparities in the background/foreground are smooth (spatially correlated) within their layers and are independent across layers. For a given segmentation,

the covariance function is

$$c(\mathbf{x}_i, \mathbf{x}_j; s) = \begin{cases} De^{-\alpha\|\mathbf{x}_i - \mathbf{x}_j\|^2} & s(\mathbf{x}_i) = s(\mathbf{x}_j) \neq \mathsf{O} \\ D\delta(\mathbf{x}_i - \mathbf{x}_j) & s(\mathbf{x}_i) = s(\mathbf{x}_j) = \mathsf{O} \\ 0 & s(\mathbf{x}_i) \neq s(\mathbf{x}_j) \end{cases} \tag{2}$$

where $D$ is the maximum disparity in the scene and $\delta$ is the Dirac delta function. The covariance of two points will be zero (i.e., the disparities are independent) unless they share the same segmentation label. Disparity is undefined within occlusion regions so these points are treated as independent with high variance to capture the noisy observations that occur here, pixels with other labels have disparities whose covariance falls off with distance engendering smoothness in the disparity map; the parameter $\alpha$ controls the smoothness and is set to $\alpha = 0.01$ for all of the experiments shown in this paper (the points $\mathbf{x}$ are measured in pixel units). It will be convenient in what follows to define the covariance for sets of points such that $c(\mathcal{X}, \mathcal{X}'; s) = C(s) \in \mathbb{R}^{n \times n'}$ where the element $C_{ij}$ is the covariance of the $i^{\text{th}}$ element of $\mathcal{X}$ and $j^{\text{th}}$ element of $\mathcal{X}'$. The prior mean is also defined according to segmentation to reflect the fact that the foreground is at greater disparity (nearer the camera) than the background

$$f(\mathbf{x}; s) = \begin{cases} 0.2D & s(\mathbf{x}) = \mathsf{B} \\ 0.8D & s(\mathbf{x}) = \mathsf{F} \\ 0.5D & s(\mathbf{x}) = \mathsf{O} \end{cases}. \tag{3}$$

Because of the independence induced by the discrete labels $s(\mathbf{x})$, we call this prior model a *switched Gaussian process*. Switching between Gaussian processes for different parts of the input space has been discussed previously by [10] in which switching was demonstrated for a 1D regression problem.

## 2.2 Stereo measurement process

A proposed disparity $d(\mathbf{x})$ is compared to the data via the *normalized sum of squared differences* (NSSD) matching cost over a region $\Omega$ (here a $5 \times 5$ pixel patch centred at the origin) using the normalized intensity is $\bar{L}(\mathbf{x}) = L(\mathbf{x}) - \frac{1}{25}\sum_{\mathbf{a}\in\Omega} L(\mathbf{x}+\mathbf{a})$ (likewise for $\bar{R}(\mathbf{x})$)

$$m(\mathbf{x}, d) = \frac{\sum_{\mathbf{a}\in\Omega}\left(\bar{L}(\mathbf{x}+\mathbf{a}) - \bar{R}(\mathbf{x}+\mathbf{a}-\mathbf{d})\right)^2}{2\sum_{\mathbf{a}\in\Omega}\left(\bar{L}^2(\mathbf{x}+\mathbf{a}) + \bar{R}^2(\mathbf{x}+\mathbf{a}+\mathbf{d})\right)}. \tag{4}$$

This cost has been shown in practice to be effective for disparity estimation [11].

To incorporate this information with the GP prior it must be expressed probabilistically. We follow the approach of [12] for this in which a parabola is fitted around the disparity with minimum score $m(\mathbf{x}, d) \approx ad^2 + bd + c$. Interpreting this as the inverse logarithm of a Gaussian distribution gives

$$d(\mathbf{x}) = \mu(\mathbf{x}) + \epsilon \quad \text{where} \quad \epsilon \sim \text{Normal}\left(0,\ v(\mathbf{x})\right) \tag{5}$$

with $\mu(\mathbf{x}) = -\frac{b}{a}$ and $v(\mathbf{x}) = \frac{1}{2a}$ being the observation mean and variance.

Given a segmentation and a set of noisy measurements at locations $\mathcal{X} = \{\mathbf{x}_i, \ldots, \mathbf{x}_n\}$, the GP can be used to predict the disparity at a new point $P(d(\mathbf{x})|\mathcal{X})$. This is a Gaussian distribution Normal $(\tilde{\mu}(\mathbf{x}),\ \tilde{v}(\mathbf{x}))$ with [9]

$$\tilde{\mu}(\mathbf{x}; s) = \mu^{\text{T}}\tilde{C}(s)^{-1}c(\mathcal{X}, \mathbf{x}; s) \quad \text{and} \quad \tilde{v}(\mathbf{x}; s) = c(\mathbf{x}, \mathbf{x}; s) - c(\mathcal{X}, \mathbf{x}; s)^{\text{T}}\tilde{C}(s)^{-1}\mathbf{c}(\mathbf{x}; s) \tag{6}$$

where $\tilde{C}(s) = c(\mathcal{X}, \mathcal{X}; s) + \text{diag}\left(v(\mathbf{x}_1), \ldots, v(\mathbf{x}_n)\right)$ and $\mu = [\mu(\mathbf{x}_1), \ldots, \mu(\mathbf{x}_n)]^{\text{T}}$.

## 2.3 Segmentation likelihood

The previous discussion has assumed that the segmentation is known, yet this will rarely be the case in practice: $s$ must therefore be inferred from the data together with the disparity. For a given set of observations, the probability that they are a sample from the GP prior is given by

$$E(\mathcal{X}) = \log P\left(\mu|s, \mathbf{v}\right) = -\left[\mu - \mathbf{f}(s)\right]^{\text{T}}\tilde{C}(s)^{-1}\left[\mu - \mathbf{f}(s)\right] - \log \det \tilde{C}(s) - \frac{n}{2}\log 2\pi. \tag{7}$$

This is the *evidence* for the parameters of the prior model and constitutes a data likelihood for the segmentation. The next section describes an algorithm that uses this quantity to infer a segmentation whilst incorporating observations.

# 3 Incremental incorporation of measurements and model selection

We propose an incremental and greedy algorithm for finding a segmentation. Measurements are incorporated one at a time and the evidence of adding the $i^{\text{th}}$ observation to each of the three segmentation layers is computed based on the preceding $i - 1$ observations and their labels. The $i^{\text{th}}$ point is labelled according to which gave the greatest evidence. The first $i - 1$ observation points $\mathcal{X}_{i-1} = \{\mathbf{x}_1, \dots, \mathbf{x}_{i-1}\}$ are partitioned according to their labelling into the mutually independent sets $\mathcal{X}_\mathsf{F}$, $\mathcal{X}_\mathsf{B}$ and $\mathcal{X}_\mathsf{O}$. Since the three segmentation layers are independent, some of the cost of computing and storing the large matrix $\tilde{C}^{-1}$ is avoided by constructing $\tilde{F}^{-1}$ and $\tilde{B}^{-1}$ instead where $\tilde{F} = c(\mathcal{X}_\mathsf{F}, \mathcal{X}_\mathsf{F})$ and $\tilde{B} = c(\mathcal{X}_\mathsf{B}, \mathcal{X}_\mathsf{B})$. Observations assigned to the occlusion layer are independent of all other points and contain no useful information. There is therefore no need to keep a covariance matrix for these.

As shown in [13], the GP framework easily facilitates incremental incorporation of observations by repeatedly updating the matrix inverse required in the prediction equations (6). For example, to add the $i^{\text{th}}$ example to the foreground (the process is identical for the background layer) compute

$$\tilde{F}_i^{-1} = \begin{bmatrix} \tilde{F}_{i-1} & \mathbf{c}(\mathcal{X}_\mathsf{F}, \mathbf{x}_i) \\ \mathbf{c}(\mathcal{X}_\mathsf{F}, \mathbf{x}_i)^\mathsf{T} & c(\mathbf{x}_i, \mathbf{x}_i) + v(\mathbf{x}) \end{bmatrix}^{-1} = \begin{bmatrix} \tilde{F}_{i-1}^{-1} + \mathbf{q}_\mathsf{F} \mathbf{q}_\mathsf{F}^\mathsf{T}/r_\mathsf{F} & \mathbf{q}_\mathsf{F} \\ \mathbf{q}_\mathsf{F}^\mathsf{T} & r_\mathsf{F} \end{bmatrix} \tag{8}$$

where

$$r_\mathsf{F}^{-1} = c(\mathbf{x}_n, \mathbf{x}_n) + v(\mathbf{x}_i) - \mathbf{c}(\mathcal{X}_\mathsf{F}, \mathbf{x})^\mathsf{T} \tilde{F}_{i-1}^{-1} \mathbf{c}(\mathcal{X}_\mathsf{F}, \mathbf{x})$$
$$\mathbf{q}_\mathsf{F} = -r_\mathsf{F} \tilde{F}_{i-1}^{-1} \mathbf{c}(\mathcal{X}_\mathsf{F}, \mathbf{x}). \tag{9}$$

Similarly, there is an incremental form for computing the evidence of a particular segmentation as $E(\mathcal{X}_i | s(\mathbf{x}_i) = j) = E(\mathcal{X}_{i-1}) + \Delta E_j(\mathbf{x}_i)$ where

$$\Delta E_j(\mathbf{x}_i) = \log(r_j) - \frac{(\mu(\mathcal{X}_j)^\mathsf{T} \mathbf{q}_j)^2}{r_j} - 2\mu(\mathbf{x}_i)\mathbf{q}_j^\mathsf{T}\mu(\mathcal{X}_j) - r_j \mu(\mathbf{x}_i)^2 - \tfrac{1}{2}\log 2\pi \tag{10}$$

By computing $\Delta E_j$ for the three possible segmentations, a new point can be greedily labelled as that which gives the greatest increase in evidence.

Algorithm 1 gives pseudo-code for the incremental incorporation of a measurement and greedy labelling. As with Gaussian Process regression in general, this algorithm scales as $\mathcal{O}(n^2)$ for storage and $\mathcal{O}(n^3)$ for time and it is therefore impractical to make an observation at every pixel for images of useful size. We propose two mechanisms to overcome this:

1. Factorize the image into several sub-images and treat each one independently. The next subsection demonstrates this when each scanline (row of pixels) is handled independently.

2. Only make measurements at a sparse subset of locations. Subsection 3.2 describes an active learning approach for identifying optimally informative observation points.

## 3.1 Independent scanline observation schedule

By handling the image pixels one row at a time, the problem becomes one-dimensional. Points are processed in order from right to left: for each point the disparity is measured as described in

---

**Algorithm 1 Add and label measurement at $\mathbf{x}_i$**

    **input** $\tilde{F}^{-1}$, $\tilde{B}^{-1}$, $\mathcal{X}_\mathsf{F}$, $\mathcal{X}_\mathsf{B}$ and $\mathcal{X}_\mathsf{O}$
    Compute matrix building blocks $r_{j \in \{\mathsf{F},\mathsf{B}\}}$ and $\mathbf{q}_{j \in \{\mathsf{F},\mathsf{B}\}}$ from (9)
    Compute change in evidence for adding to each layer $\Delta E_{j \in \{\mathsf{F},\mathsf{B},\mathsf{O}\}}$ from (10)
    Label point $s(\mathbf{x}_i) = \arg\max_{j \in \{\mathsf{F},\mathsf{B},\mathsf{O}\}} \Delta E_j(\mathbf{x}_i)$
    Add point to set $\mathcal{X}_{s(\mathbf{x}_i)} \leftarrow \mathcal{X}_{s(\mathbf{x}_i)} \cup \mathbf{x}_i$
    **if** $s(\mathbf{x}_i) \in \mathsf{F} \cup \mathsf{B}$ **then**
        Update matrix $\tilde{F}^{-1}$ or $\tilde{B}^{-1}$ as (8)
    **end if**
    $i = i + 1$
    **return** $\tilde{F}^{-1}$, $\tilde{B}^{-1}$, $\mathcal{X}_\mathsf{F}$, $\mathcal{X}_\mathsf{B}$ and $\mathcal{X}_\mathsf{O}$

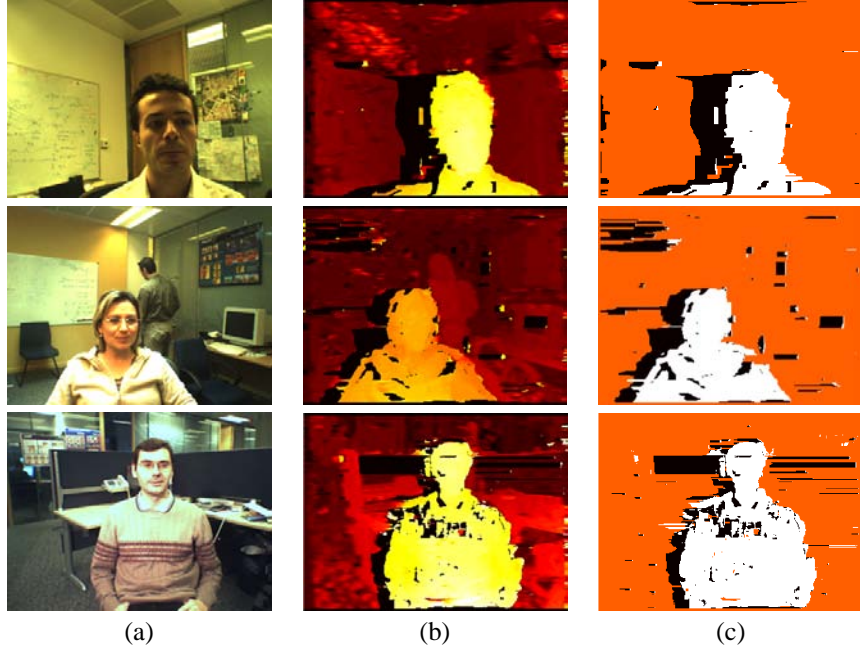

|       |       |       |
|:-----:|:-----:|:-----:|
|  (a)  |  (b)  |  (c)  |

Figure 2: **Scanline predictions.** *Disparity and segmentation maps inferred by treating each scanline independently.* (a) $320 \times 240$ *pixel left input images.* (b) *Mean predicted disparity map* $\tilde{\mu}(\mathbf{x})$. (c) *Inferred segmentation* $s(\mathbf{x})$ *with* $\mathsf{F}$ = *white*, $\mathsf{B}$ = *grey (orange) and* $\mathsf{O}$ = *black.*

Sec. 2.2 and incorporated/labelled according to Algorithm 1. In this setting there are constraints on which labels may be neighbours along a scanline. Fig. 1 shows the segmentation for a typical image from which it can be seen that, moving horizontally from right to left, the only "legal" transitions in segmentation are $\mathsf{B} \rightarrow \mathsf{F}$, $\mathsf{F} \rightarrow \mathsf{O}$ and $\mathsf{O} \rightarrow \mathsf{B}$. Algorithm 1 is therefore modified to consider legal segmentations only.

Fig. 2 shows some results of this approach. Both the disparities and segmentation are, subjectively, accurate however there are a number of "streaky" artifacts caused by the fact that there is no vertical sharing of information. There are also a number of artifacts where an incorrect segmentation label has been assigned; in many cases this is where a point in the foreground or background has been labelled as occluded because there is no texture in that part of an image and measurements made for such points have an high variance. The occlusion class could therefore be more accurately described as a general outlier category.

## 3.2    Active selection of sparse measurement locations

As shown above, our GP model scales badly with the number of observations. The previous subsection used measurements at all locations by treating each scanline as independent, however a shortcoming of this approach is that no information is propagated vertically, introducing streaky artifacts and reducing the model's ability to reason about occlusions and discontinuities.

Rather than introduce artificial independencies, the observation schedule in this section copes with the $\mathcal{O}(n^3)$ scaling by making measurements at only a sparse set of locations. Obvious ways of implementing this include choosing $n$ locations either at random or in a grid pattern, however these fail to exploit information that can be readily obtained from both the image data and the current predictions made by the model. Hence, we propose an *active* approach, similar to that in [14]: given the first $i - 1$ observations, observe the point which maximally reduces the entropy of the GP [8]

$$\Delta H(\mathbf{x}) = H\big(P(d|\mathcal{X}_{-1})\big) - H\big(P(d|\mathcal{X}_{-1} \cup \mathbf{x})\big) = -\tfrac{1}{2}\log\det\Sigma + \tfrac{1}{2}\log\det\Sigma' + \text{const.} \quad (11)$$

where $\Sigma$ and $\Sigma'$ are the posterior covariances of the GP over all points in the image before and after making an observation at $\mathbf{x}$. To compute the entire posterior for each observation would be prohibitively expense; instead we approximate it by the product of the marginal distributions at each

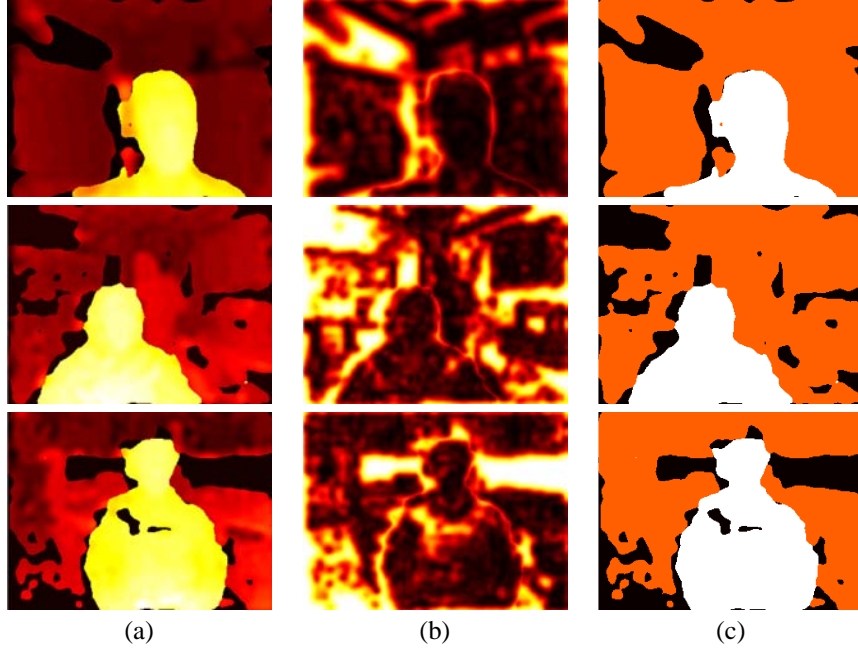

|       |       |       |
|:-----:|:-----:|:-----:|
| (a)   | (b)   | (c)   |

Figure 3: **Predictions after sparse active observation schedule.** *This figure shows the predictions made by the GP model with observations at 1000 image locations for the images used in Fig. 2.* (a) *Mean predicted disparity $\tilde{\mu}(\mathbf{x})$;* (b) *Predictive uncertainty $\tilde{v}(\mathbf{x})$;* (c) *Inferred segmentation.*

point (i.e., ignore off-diagonal elements in $\Sigma$) which gives $\Delta H(\mathbf{x}) \approx \frac{1}{2}\left(\log \tilde{v}(\mathbf{x}) - \log v(\mathbf{x})\right)$ where $\tilde{v}(\mathbf{x})$ is the predicted variance from (6) and $v(\mathbf{x})$ is the measurement variance. Since the logarithm is monotonic, an equivalent utility function is used:

$$U\left(\mathbf{x}|\mathcal{X}_{i-1}\right) = \frac{\tilde{v}(\mathbf{x})}{v(\mathbf{x})}. \tag{12}$$

Here the numerator drives the system to make observations at points with greatest predictive uncertainty. However, this is balanced by the denominator to avoid making observations at points where there is no information to be obtained from the data (e.g., the textureless regions in Fig. 2). To initialize the active algorithm, 64 initial observations are made in a evenly spaced grid over the image. Following this, points are selected using the utility function (12) and incorporated into the GP model using Algorithm 1.

Predicting disparity in the scanline factorization was straightforward because a segmentation label had been assigned to every pixel. With sparse measurements, only the observation points have been labelled and to predict disparity at an arbitrary location a segmentation label must also be inferred. Our simple strategy for this is to label a point according to which gives the least predictive variance, i.e.:

$$s(\mathbf{x}) = \arg\min_{j \in \{F,B,O\}} \tilde{v}(\mathbf{x}; s(\mathbf{x}) = j). \tag{13}$$

Fig. 3 shows the results of using this active observation schedule with $n = 1000$ for the images of Fig. 2. As expected, by restoring 2D spatial coherence the results are smoother and have none of the streaky artifacts induced by the scanline factorization. Despite observing only 1.3% of the points used by the scanline factorization, the active algorithm has still managed to capture the important features in the scenes. Fig. 4a shows the locations of the $n$ observation points; the observations are clustered around the boundary of the foreground object in an attempt to minimize the uncertainty at discontinuities/occlusions; the algorithm is dedicating its computational resources to the parts of the image which are most interesting, important and informative. Fig. 4b demonstrates further the benefits of selecting the points in an active framework compared to taking points at random.

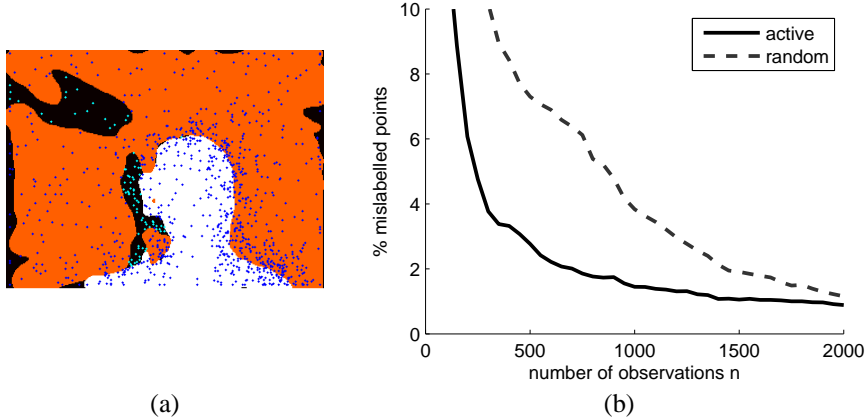

(a)                  (b)

Figure 4: **Advantage of active point selection.** (a) *The inferred segmentation from Fig. 3 with spots (blue) corresponding to observation locations selected by the active criterion.* (b) *This plot compares the accuracy of the segmentation against the number of sparse observations when the observation locations are chosen at random and using our active schedule. Accuracy is measured as the percentage of mislabelled pixels compared to a hand-labelled ground truth segmentation. The active strategy achieves better accuracy with fewer observations.*

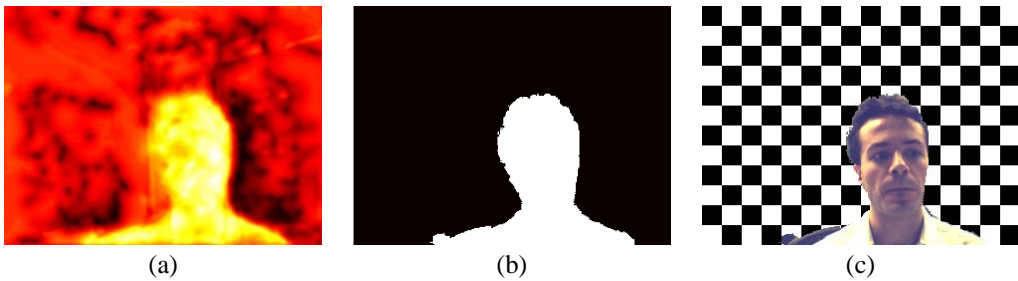

(a)             (b)             (c)

Figure 5: **Improved segmentation by fusion with colour.** (a) *Pixel-wise energy term $V(\mathbf{x})$ combining segmentation predictions from both the switched GP posterior and a colour model;* (b) *Segmentation returned by the Viterbi algorithm. This contains 0.5% labelling errors by area.* (c) *Inferred foreground image pixels.*

## 3.3 Adding colour information

The best segmentation accuracies using stereo information alone are around 1% labelling errors (with $n \geq 1000$). In [5], superior segmentation results are achieved by incorporating colour information. We do the same here by computing a foreground "energy" $V(\mathbf{x})$ at each location based on the variances predicted by the foreground/background layers and a known colour distribution $P\big(\mathsf{F}|L_c(\mathbf{x})\big)$ where $L_c(\mathbf{x})$ is the RGB colour of the left image at $\mathbf{x}$:

$$V(\mathbf{x}) = \log \tilde{v}(\mathbf{x}; s(\mathbf{x}) = \mathsf{B}) - \log \tilde{v}(\mathbf{x}; s(\mathbf{x}) = \mathsf{F}) - \log P\big(\mathsf{F}|L_c(\mathbf{x})\big). \tag{14}$$

We represent the colour distribution using a $10 \times 10 \times 10$ bin histogram in red-green-blue colour space. Fig. 5a shows this energy for the first image in Fig. 2. As in [5], we treat each scanline as a binary HMM and use the Viterbi algorithm to find a segmentation. A result of this is shown in Fig. 5c which contains 0.58% erroneous labels. This is comparable to the errors in [5] which are around 0.25% for this image. We suspect that our result can be improved with a more sophisticated colour model.

## 4 Discussion

We have proposed a Gaussian process model for disparity, switched by a latent segmentation variable. We call this a *switched Gaussian process* and have proposed an incremental greedy algorithm

for fitting this model to data and inferring a segmentation. We have demonstrated that by using a sparse model with points selected according to an active learning criterion, an accuracy can be achieved that is comparable to the state of the art [5].

We believe there are four key strengths to this probabilistic framework:

**Flexibility** The incremental nature of the algorithm makes it possible to set the number of observations $n$ according to time or quality constraints.

**Extensibility** This method is probabilistic so fusion with other sources of information is possible (e.g., laser range scanner on a robot).

**Efficiency** For small $n$, this approach is very fast ( 30ms per pair of images for $n = 200$ on a 3GHz PC). However, higher quality results require $n > 1000$ observations which reduces the execution speed to a few seconds per image.

**Accuracy** We have shown that (for large $n$) this technique achieves an accuracy comparable to the state of the art.

Future work will investigate the use of approximate techniques to overcome the $\mathcal{O}(n^3)$ scaling problem [15]. The framework described in this paper can operate at real time for low $n$, however any technique that combats the scaling will allow higher accuracy for the same execution time. Also, improving the approximation to the likelihood in (5), e.g., by expectation propagation [16], may increase accuracy.

## References

[1] D. Comaniciu and P. Meer. Robust analysis of feature spaces: color image sgementation. In *Proc. Conf. Computer Vision and Pattern Recognition*, pages 750–755, 1997.

[2] Y. Ohta and T. Kanade. Stereo by intra- and inter-scanline search using dynamic programming. *IEEE Trans. on Pattern Analysis and Machine Intelligence*, 7(2):139–154, 1985.

[3] D. Geiger, B. Ladendorf, and A. Yuille. Occlusions and binocular stereo. *Int. J. Computer Vision*, 14:211–226, 1995.

[4] V. Kolmogorov and R. Zabih. Computing visual correspondence with occlusions using graph cuts. In *Proc. Int. Conf. Computer Vision*, 2001.

[5] V. Kolmogorov, A. Criminisi, A. Blake, G. Cross, and C. Rother. Bi-layer segmentation of binocular stereo video. In *Proc. Conf. Computer Vision and Pattern Recognition*, 2005.

[6] F. Sinz, J. Quiñonero-Candela, G.H. Bakir, C.E. Rasmussen, and M.O. Franz. Learning depth from stereo. In *Pattern Recognition, Proc. 26th DAGM Symposium*, pages 245–252, 2004.

[7] R. Hartley and A. Zisserman. *Multiple View Geometry*. Cambridge University Press, 2000.

[8] D.J.C. MacKay. Information-based objective functions for active data selection. *Neural Computation*, 4(4):589–603, 1992.

[9] C.E. Rasmussen and C.K.I. Williams. *Gaussian Processes for Machine Learning*. MIT Press, 2006.

[10] A. Storkey. Gaussian processes for switching regimes. In *Proc. ICANN*, 1998.

[11] D. Scharstein and R. Szeliski. A taxonomy and evaluation of desnse two-frame stereo correspondence algorithms. *Int. J. Computer Vision*, 47(1):7–42, 2002.

[12] L. Matthies, R. Szeliski, and T. Kanade. Incremental estimation of dense depth maps from image sequences. In *Proc. Conf. Computer Vision and Pattern Recognition*, 1988.

[13] M. Gibbs and D.J.C. MacKay. Efficient implementation of gaussian processes. Technical report, University of Cambridge, 1997.

[14] M. Seeger, C.K.I. Williams, and N. Lawrence. Fast forward selection to speed up sparse gaussian process regression. In *Proc. AI-STATS*, 2003.

[15] J. Quiñonero-Candela and C.E. Rasmussen. A unifying view of sparse approximate Gaussian process regression. *J. Machine Learning Research*, 6:1939–1959, 2005.

[16] T.P. Minka. Expectation propagation for approximate Bayesian inference. In *Proc. UAI*, pages 362–369, 2001.
